# Sparse Filtering

**Jiquan Ngiam, Pang Wei Koh, Zhenghao Chen, Sonia Bhaskar, Andrew Y. Ng**
Computer Science Department, Stanford University
`{jngiam,pangwei,zhenghao,sbhaskar,ang}@cs.stanford.edu`

## Abstract

Unsupervised feature learning has been shown to be effective at learning representations that perform well on image, video and audio classification. However, many existing feature learning algorithms are hard to use and require extensive hyperparameter tuning. In this work, we present *sparse filtering*, a simple new algorithm which is efficient and only has one hyperparameter, the number of features to learn. In contrast to most other feature learning methods, sparse filtering does not explicitly attempt to construct a model of the data distribution. Instead, it optimizes a simple cost function – the sparsity of $\ell_2$-normalized features – which can easily be implemented in a few lines of MATLAB code. Sparse filtering scales gracefully to handle high-dimensional inputs, and can also be used to learn meaningful features in additional layers with greedy layer-wise stacking. We evaluate sparse filtering on natural images, object classification (STL-10), and phone classification (TIMIT), and show that our method works well on a range of different modalities.

## 1 Introduction

Unsupervised feature learning has recently emerged as a viable alternative to manually designing feature representations. In many audio [1, 2], image [3, 4], and video [5] tasks, learned features have matched or outperformed features specifically designed for such tasks. However, many current feature learning algorithms are hard to use because they require a good deal of hyperparameter tuning. For example, the sparse RBM [6, 7] has up to half a dozen hyperparameters and an intractable objective function, making it hard to tune and monitor convergence.

In this work, we present *sparse filtering*, a new feature learning algorithm which is easy to implement and essentially hyperparameter-free. Sparse filtering is efficient and scales gracefully to handle large input dimensions. In contrast, it is typically computationally expensive to run straightforward implementations of many other feature learning algorithms on large inputs.

Sparse filtering works by optimizing exclusively for sparsity in the feature distribution. A key idea in our method is avoiding explicit modeling of the data distribution; this gives rise to a simple formulation and permits efficient learning. As a result, our method can be implemented in a few lines of MATLAB code[1] and works well with an off-the-shelf function minimizer such as L-BFGS.

Moreover, the hyperparameter-free approach means that sparse filtering works well on a range of data modalities without the need for specific tuning on each modality. This allows us to easily learn feature representations that are well-suited for a variety of tasks, including object classification and phone classification.

Table 1. Comparison of tunable hyperparameters in various feature learning algorithms.

| Algorithm | Tunable hyperparameters |
|---|---|
| Our Method (Sparse Filtering) | # features |
| ICA | # features |
| Sparse Coding | # features, sparsity penalty, mini-batch size |
| Sparse Autoencoders | # features, target activation, weight decay, sparsity penalty |
| Sparse RBMs | # features, target activation, weight decay, sparsity penalty, learning rate, momentum |

## 2   Unsupervised feature learning

Traditionally, feature learning methods have largely sought to learn models that provide good approximations of the true data distribution; these include denoising autoencoders [8], restricted Boltzmann machines (RBMs) [6, 7], (some versions of) independent component analysis (ICA) [9, 10], and sparse coding [11], among others.

These feature learning approaches have been successfully used to learn good feature representations for a wide variety of tasks [1, 2, 3, 4, 5]. However, they are also often challenging to implement, requiring the tuning of various hyperparameters; see Table 1 for a comparison of tunable hyperparameters in several popular feature learning algorithms. Good settings for these hyperparameters can vary widely from task to task, and can sometimes result in a drawn-out development process. Though ICA has only one tunable hyperparameter, it scales poorly to large sets of features or large inputs.[2]

In this work, our goal is to develop a simple and efficient feature learning algorithm that requires minimal tuning. To this end, we only focus on a few key properties of our features – population sparsity, lifetime sparsity, and high dispersal – without explicitly modeling the data distribution.

While learning a model for the data distribution is desirable, it can complicate learning algorithms: for example, sparse RBMs need to approximate the log-partition function's gradient in order to optimize for the data likelihood, while sparse coding needs to run relatively expensive inference at each iteration to find the coefficients of the active bases. The relative weightage of a data reconstruction term versus a sparsity-inducing term is also often a hyperparameter that needs to be tuned.

## 3   Feature distributions

The feature learning methods discussed in the previous section can all be viewed as generating particular feature distributions. For instance, sparse coding represents each example using a few non-zero coefficients (features). A feature distribution oriented approach can provide insights into designing new algorithms based on optimizing for desirable properties of the feature distribution.

For clarity, let us consider a feature distribution matrix over a finite dataset, where each row is a feature, each column is an example, and each entry $f_j^{(i)}$ is the activity of feature $j$ on example $i$. We assume that the features are generated through some deterministic function of the examples.

We consider the following as desirable properties of the feature distribution:

**Sparse features per example (*Population Sparsity*).** Each example should be represented by only a few active (non-zero) features. Concretely, for each column (one example) in our feature matrix, $f^{(i)}$, we want a small number of active elements. For example, an image can be represented by a description of the objects in it, and while there are many possible objects that can appear, only a few are typically present at a single time. This notion is known as population sparsity [13, 14] and is considered a principle adopted by the early visual cortex as an efficient means of coding.

**Sparse features across examples (*Lifetime Sparsity*).** Features should be discriminative and allow us to distinguish examples; thus, each feature should only be active for a few examples. This means that each row in the feature matrix should have few non-zero elements. This property is known as lifetime sparsity [13, 14].

**Uniform activity distribution (*High Dispersal*).** For each row, the distribution should have similar statistics to every other row; no one row should have significantly more "activity" than the other rows. Concretely, we consider the mean squared activations of each feature obtained by averaging the squared values in the feature matrix across the columns (examples). This value should be roughly the same for all features, implying that all features have similar contributions. While high dispersal is not strictly necessary for good feature representations, we found that enforcing high dispersal prevents degenerate situations in which the same features are always active [14]. For overcomplete representations, high dispersal translates to having fewer "inactive" features. As an example, principle component analysis (PCA) codes do not generally satisfy high dispersal since the codes that correspond to the largest eigenvalues are almost always active.

These properties of feature distributions have been explored in the neuroscience literature [9, 13, 14, 15]. For instance, [14] showed that population sparsity and lifetime sparsity are not necessarily correlated. We note that the characterization of neural codes have conventionally been expressed as properties of the *feature distribution*, rather than as a way of modeling the *data distribution*.

Many feature learning algorithms include these objectives. For example, the sparse RBM [6] works by constraining the expected activation of a feature (over its lifetime) to be close to a target value. ICA [9, 10] has constraints (e.g., each basis has unit norm) that normalize each feature, and further optimizes for the lifetime sparsity of the features it learns. Sparse autoencoders [16] also explicitly optimize for lifetime sparsity.

On the other hand, clustering-based methods such as $k$-means [17] can be seen as enforcing an extreme form of population sparsity where each cluster centroid corresponds to a feature and only one feature is allowed to be active per example. "Triangle" activation functions, which essentially serve to ensure population sparsity, have also been shown to obtain good classification results [17]. Sparse coding [11] is also typically seen as enforcing population sparsity.

In this work, we use the feature distribution view to derive a simple feature learning algorithm that solely optimizes for *population sparsity* while enforcing *high dispersal*. In our experiments, we found that realizing these two properties was sufficient to allow us to learn overcomplete representations; we also argue later that these two properties are jointly sufficient to ensure lifetime sparsity.

## 4 Sparse filtering

In this section, we will show how the sparse filtering objective captures the aforementioned principles. Consider learning a function that computes linear features for every example. Concretely, let $f_j^{(i)}$ represent the $j^{th}$ feature value (rows) for the $i^{th}$ example (columns), where $f_j^{(i)} = \mathbf{w_j}^T \mathbf{x}^{(i)}$. Our method simply involves first normalizing the feature distribution matrix by rows, then by columns and finally summing up the absolute value of all entries.

Specifically, we first normalize each feature to be equally active by dividing each feature by its $\ell_2$-norm across all examples: $\tilde{\mathbf{f}}_\mathbf{j} = \mathbf{f_j}/\|\mathbf{f_j}\|_2$. We then normalize these features per example, so that they lie on the unit $\ell_2$-ball, by computing $\hat{\mathbf{f}}^{(\mathbf{i})} = \tilde{\mathbf{f}}^{(\mathbf{i})}/\|\tilde{\mathbf{f}}^{(\mathbf{i})}\|_2$. The normalized features are optimized for sparseness using the $\ell_1$ penalty. For a dataset of $M$ examples, this gives us the sparse filtering objective (Eqn. 1):

$$\text{minimize} \quad \sum_{i=1}^{M} \left\| \hat{\mathbf{f}}^{(\mathbf{i})} \right\|_1 = \sum_{i=1}^{M} \left\| \frac{\tilde{\mathbf{f}}^{(\mathbf{i})}}{\|\tilde{\mathbf{f}}^{(\mathbf{i})}\|_2} \right\|_1 . \tag{1}$$

### 4.1 Optimizing for population sparsity

The term $\|\hat{\mathbf{f}}^{(\mathbf{i})}\|_1 = \left\| \frac{\tilde{\mathbf{f}}^{(\mathbf{i})}}{\|\tilde{\mathbf{f}}^{(\mathbf{i})}\|_2} \right\|_1$ measures the *population sparsity* of the features on the $i^{th}$ example. Since the normalized features $\hat{\mathbf{f}}^{(\mathbf{i})}$ are constrained to lie on the unit $\ell_2$-ball, this objective is

minimized when the features are sparse (Fig. 1-Left), which corresponds to being close to the axes. Conversely, an example which has similar values for every feature would incur a high penalty.

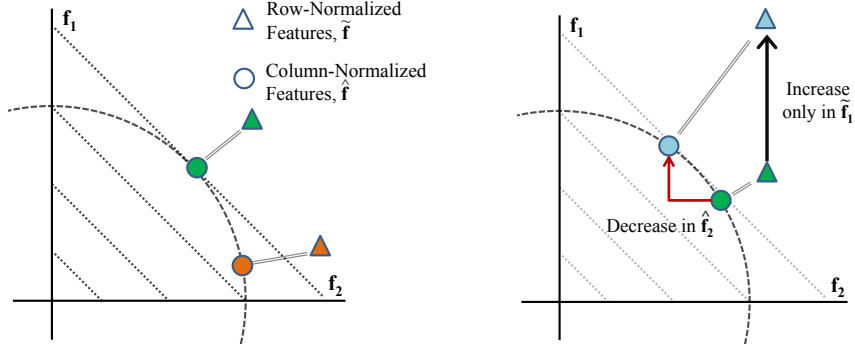

Figure 1: Left: Sparse filtering showing two features ($f_1$, $f_2$) and two examples (red and green). Each example is first projected onto the $\ell_2$-ball and then optimized for sparseness. The $\ell_2$-ball is shown together with level sets of the $\ell_1$-norm. Notice that the sparseness of the features (in the $\ell_1$ sense) is maximized when the examples are on the axes. Right: Competition between features due to normalization. We show one example where only $f_1$ is increased. Notice that even though only $f_1$ is increased, the normalized value of the second feature, $\hat{f}_2$ decreases.

One property of normalizing features is that it implicitly introduces competition between features. Notice that if only one component of $\mathbf{f}^{(\mathbf{i})}$ is increased, all the other components $\hat{f}_j^{(i)}$ will decrease because of the normalization (Fig. 1-Right). Similarly, if only one component of $\mathbf{f}^{(\mathbf{i})}$ is decreased, all other components will increase. Since we are minimizing $\|\hat{\mathbf{f}}^{(\mathbf{i})}\|_1$, the objective encourages the normalized features, $\hat{\mathbf{f}}^{(\mathbf{i})}$, to be sparse and mostly close to zero. Putting this together with the normalization, this means that some features in $\mathbf{f}^{(\mathbf{i})}$ have to be large while most of them are small (close to zero). Therefore, the objective optimizes for population sparsity.

The formulation above is closely related to the Treves-Rolls [14, 18] measure of population/lifetime sparsity: $s^{(i)} = \left[\sum_j \tilde{f}_j^{(i)}/F\right]^2 / \left[\sum_j (\tilde{f}_j^{(i)})^2/F\right]$, where $F$ is the total number of features. This measure is commonly used to characterize the sparsity of neuron activations in the brain. In particular, our proposed formulation can be viewed as a re-scaling of the square-root of this measure.

## 4.2 Optimizing for high dispersal

Recall that for high dispersal we want every feature to be equally active. Specifically, we want the mean squared activation of each feature to be roughly equal. In our formulation of sparse filtering, we first normalize each feature so that they are equally active by dividing each feature by its norm across the examples: $\tilde{\mathbf{f}}_{\mathbf{j}} = \mathbf{f}_{\mathbf{j}}/\|\mathbf{f}_{\mathbf{j}}\|_2$. This has the same effect as constraining each feature to have the same expected squared value, $E_{\mathbf{x}^{(\mathbf{i})} \sim D}[(f_j^{(i)})^2] = 1$, thus enforcing high dispersal.

## 4.3 Optimizing for lifetime sparsity

We found that optimizing for population sparsity and enforcing high dispersal led to lifetime sparsity in our features. To understand how lifetime sparsity is achieved, first notice that a feature distribution which is population sparse must have many non-active (zero) entries in the feature distribution matrix. Since these features are highly dispersed, these zero entries (and also the non-zero entries) are approximately evenly distributed among all the features. Therefore, every feature must have a significant number of zero entries and be lifetime sparse. This implies that optimizing for population sparsity and high dispersal is sufficient to define a good feature distribution.

## 4.4   Deep sparse filtering

Since the sparse filtering objective is agnostic about the method which generates the feature matrix, one is relatively free to choose the feedforward network that computes the features. It is thus possible to use more complex non-linear functions (e.g., $f_j^{(i)} = \log(1 + (\mathbf{w_j^T x^{(i)}})^2))$, or even multi-layered networks, when computing the features. In this way, sparse filtering presents itself as a natural framework for training deep networks.

Training a deep network with sparse filtering can be achieved using the canonical greedy layerwise approach [7, 19]. In particular, after training a single layer of features with sparse filtering, one can compute the normalized features $\hat{\mathbf{f}}^{(i)}$ and then use these as input to sparse filtering for learning another layer of features. In practice, we find that greedy layer-wise training with sparse filtering learns meaningful representations on the next layer (Sec. 5.2).

## 5   Experiments

In our experiments, we adopted the soft-absolute function $f_j^{(i)} = \sqrt{\epsilon + (\mathbf{w_j^T x^{(i)}})^2} \approx |\mathbf{w_j^T x^{(i)}}|$ as our activation function, setting $\epsilon = 10^{-8}$, and used an off-the-shelf L-BFGS [20] package to optimize the sparse filtering objective until convergence.

## 5.1   Timing and scaling up

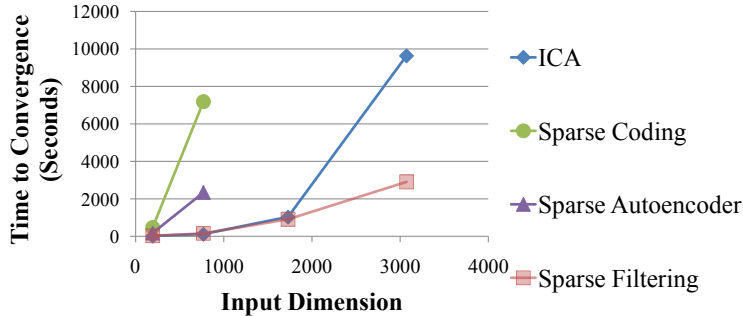

Figure 2: Timing comparisons between sparse coding, ICA, sparse autoencoders and sparse filtering over different input sizes.

In this section, we examine the efficiency of the sparse filtering algorithm by comparing it against ICA, sparse coding, and sparse autoencoders. We compared the convergence of each algorithm by measuring the relative change in function value over each iteration of the algorithm, stopping when this change dropped below a preset threshold. We performed experiments using 10,000 color image patches with varying image sizes to evaluate the efficiency and scalability of the methods. For each image size, we learned a complete set of features (i.e., equal to the number of input dimensions). We implemented sparse autoencoders as described in Coates et al. [17]. For sparse coding, we used code from [2], as it is fairly optimized and easy to modify.

For smaller image dimensions of sizes $8 \times 8$ (192-dimensional inputs since our images have 3 color channels) and $16 \times 16$ (768-dimensional inputs), we found that the algorithms generally performed similarly in terms of efficiency. However, with $32 \times 32$ image patches (3072-dimensional inputs), sparse coding, sparse autoencoders and ICA were significantly slower to converge than sparse filtering (Fig. 2). For ICA, each iteration of the algorithm (FastICA [12]) requires orthogonalizing the bases learned; since the cost of orthogonalization is cubic in the number of features, the algorithm can be very slow when the number of features is large. For sparse coding, as the number of features increased, it took significantly longer to solve the $\ell_1$-regularized least squares problem for finding the coefficients.

We obtained an overall speedup of at least 4x over sparse coding and ICA when learning features from $32 \times 32$ image patches. In contrast to ICA, optimizing the sparse filtering objective does not require the expensive cubic-time whitening step. For the larger input dimensions, sparse coding and sparse autoencoders did not converge in a reasonable time (<3 hours).

## 5.2 Natural images

In this section, we applied sparse filtering to learn features off 200,000 randomly sampled patches (16x16) from natural images [9]. The only preprocessing done before feature learning was to subtract the mean of each image patch from itself (i.e., removing the DC component).

The first layer of features learned by sparse filtering corresponded to Gabor-like edge detectors, similar to those learned by standard sparse feature learning methods [6, 9, 10, 11, 16]. More interestingly, when we learned a second layer of features using greedy layer-wise stacking on the features produced by the first layer, it discovers meaningful features that pool the first layer features (Fig. 3). We highlight that the second layer of features were learned using the same algorithm without any tuning or preprocessing of the data. While recent work by [21, 22] has also been able to learn meaningful second layer features, our method is simpler to implement, fast to run and does not require time-consuming tuning of hyperparameters.

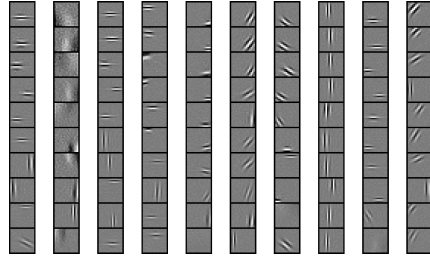

Figure 3: Learned pooling units in a second layer using sparse filtering. We show the most strongly connected first layer units for each second layer unit; each column corresponds to a second layer unit.

## 5.3 STL-10 object classification

Table 2. Classification accuracy on STL-10.

| Method | Accuracy |
|---|---|
| Raw Pixels [17] | $31.8\% \pm 0.63\%$ |
| ICA (Complete) | $48.0\% \pm 1.47\%$ |
| K-means (Triangle) [17] | $51.5\% \pm 1.73\%$ |
| Random Weight Baseline | $50.2\% \pm 1.08\%$ |
| **Our Method** | $\mathbf{53.5\% \pm 0.53\%}$ |

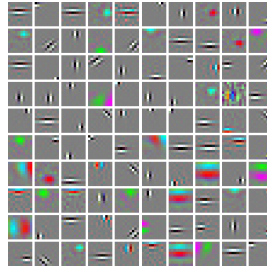

Figure 4: A subset of the learned filters from $10 \times 10$ patches extracted from the STL dataset.

We also evaluated the performance of our model on an object classification task. We used the STL-10 dataset [17] which consists of a unsupervised training set of 100,000 images, a supervised training set of 10 training folds, each with 500 training instances, and a test set of 8,000 test instances. Each instance is a $96 \times 96$ RGB image from 1 of 10 object categories.

To obtain features from the large image, we followed the protocol of [17]: features were extracted densely from all locations in each image and later pooled into quadrants. Supervised training was carried out by training a linear SVM on this representation of the training set where $C$ and the receptive field size were chosen by hold-out cross-validation. We obtain a test set accuracy of 53.5% $\pm 0.53\%$ with features learned from $10 \times 10$ patches. For a fair comparison, the number of features learnt was also set to be consistent with the number of features used by [17].

In accordance with the recommended STL-10 testing protocol [17], we performed supervised training on each of the 10 supervised training folds and reported the mean accuracy on the full test set along with the standard deviation across the 10 training folds (Table 2).

In order to show the effects of feature learning, we include a comparison to a random weight baseline of our method. For the baseline, we keep the basic architecture (e.g., the divisive normalization), but fill the entries of the weight matrix $W$ by sampling random values. Random weight baselines have been shown to perform remarkably well on a variety of tasks [23], and provide a means of distinguishing the effect of our divisive normalization scheme versus the effect of feature learning.

## 5.4 Phone classification (TIMIT)

Table 3. Test accuracy for phone classification using features learned from MFCCs.

| Method | | Accuracy |
|---|---|---|
| ICA | SVM (Linear) | 57.3% |
| MFCC | SVM (Linear) | 67.2% |
| Sparse Coding | SVM (Linear) | 76.8% |
| Our Method | SVM (Linear) | 75.7% |
| MFCC | SVM (RBF) | 80.4% |
| MFCC+ICA | SVM (RBF) | 78.3% |
| MFCC+Sparse Coding | SVM (RBF) | 80.1% |
| **MFCC+Our Method** | **SVM (RBF)** | **80.5%** |
| HMM [24] | | 78.6% |
| Large Margin GMM (LMGMM) [25] | | 78.9% |
| CRF [26] | | 79.2% |
| MFCC+CDBN [2] | | 80.3% |
| **Hierarchical LMGMM [27]** | | **81.3%** |

To evaluate the model's ability to work with a range of data modalities, we further evaluated the ability of our models to do 39-way phone classification on the TIMIT dataset [28]. As with [2, 29], our dataset comprised 132,833 training phones and 6,831 testing phones. Following standard approaches [2, 24, 25, 26, 27, 29], we first extracted 13 mel-frequency cepstral coefficients (MFCCs) and augmented them with the first and second order derivatives. Using sparse filtering, we learned 256 features from contiguous groups of 11 MFCC frames. For comparison, we also learned sets of 256 features in a similar way using sparse coding [11, 30] and ICA [12]. A fixed-length feature vector was formed from each example using the protocol described in [29].[3]

To evaluate the relative performances of the different feature sets (MFCC, ICA, sparse coding and sparse filtering), we used a linear SVM, choosing the regularization coefficient $C$ by cross-validation on the development set. We found that the features learned using sparse filtering outperformed MFCC features alone and ICA features; they were also competitive with sparse coding and faster to compute. Using an RBF kernel [31] gave performances competitive with state-of-the-art methods when MFCCs were combined with learned sparse filtering features (Table 3). In contrast, concatenating ICA and sparse coding features with MFCCs resulted in decreased performance when compared to MFCCs alone.

While methods such as the HMM and LMGMM were included in Table 3 to provide context, we note that these methods use pipelines that are more complex than straightforwardly applying a SVM. Indeed, these pipelines are built on top of feature representations that can be derived from a variety of sources, including sparse filtering. We thus see these methods as being complementary to sparse filtering.

## 6 Discussion

### 6.1 Connections to divisive normalization

Our formulation of population sparsity in sparse filtering is closely related to divisive normalization [32] – a process in early visual processing in which a neuron's response is divided by a (weighted)

sum of the responses of neighboring neurons. Divisive normalization has previously been found [33, 34] to be useful as part of a multi-stage object classification pipeline. However, it was introduced as a processing stage [33], rather than a part of unsupervised feature learning (pretraining). Conversely, sparse filtering uses divisive normalization as an integral component of the feature learning process to introduce competition between features, resulting in population sparse representations.

## 6.2    Connections to ICA and sparse coding

The sparse filtering objective can be viewed as a normalized version of the ICA objective. In ICA [12], the objective is to minimize the response of linear filters (e.g., $\|W\mathbf{x}\|_1$), subject to the constraint that the filters are orthogonal to each other. The orthogonality constraint results in set of diverse filters. In sparse filtering, we replace the objective with a normalized sparsity penalty, where the response of filters are divided by the norm of the all the filters ($\|W\mathbf{x}\|_1/\|W\mathbf{x}\|_2$). This introduces competition between the filters and thus removes the need for orthogonalization.

Similarly, one can apply the normalization idea to the sparse coding framework. In particular, sparse filtering resembles the $\frac{\ell_1}{\ell_2}$ sparsity penalty that has been used in non-negative matrix factorization [35]. Thus, instead of the usual $\ell_1$ penalty that is used in conjunction with sparse coding (i.e., $\|s\|_1$), one can instead use a normalized penalty (i.e., $\|s\|_1/\|s\|_2$). This normalized penalty is scale invariant and can be more robust to variations in the data.

## Footnotes

[1]We have included a complete MATLAB implementation of sparse filtering in the supplementary material.

[2]ICA is unable to learn overcomplete feature representations unless one resorts to extremely expensive approximate orthogonalization algorithms [12]. Even when learning complete feature representations, it still requires an expensive orthogonalization step at every iteration.

[3]We found that spliting the features into positive and negative components improved performance slightly.

## References

[1] G. E. Dahl, M. Ranzato, A. Mohamed, and G. E. Hinton. Phone recognition with the mean-covariance restricted Boltzmann machine. In *NIPS*. 2010.

[2] H. Lee, Y. Largman, P. Pham, and A. Y. Ng. Unsupervised feature learning for audio classification using convolutional deep belief networks. In *NIPS*. 2009.

[3] J. Yang, K. Yu, Y. Gong, and T. Huang. Linear spatial pyramid matching using sparse coding for image classification. In *CVPR*, 2009.

[4] M.A. Ranzato, F. J. Huang, Y.-L. Boureau, and Y. LeCun. Unsupervised learning of invariant feature hierarchies with applications to object recognition. In *CVPR*, 2007.

[5] Q. V. Le, W. Y. Zou, S. Y. Yeung, and A. Y. Ng. Learning hierarchical spatio-temporal features for action recognition with independent subspace analysis. In *CVPR*, 2011.

[6] H. Lee, C. Ekanadham, and A.Y. Ng. Sparse deep belief net model for visual area v2. In *NIPS*, 2008.

[7] G. E. Hinton, S. Osindero, and Y.W. Teh. A fast learning algorithm for deep belief nets. *Neural Computation*, 18(7):1527–1554, 2006.

[8] P. Vincent, H. Larochelle, Y. Bengio, and P. A. Manzagol. Extracting and composing robust features with denoising autoencoders. In *ICML*, 2008.

[9] J. H. van Hateren and A. van der Schaaf. Independent component filters of natural images compared with simple cells in primary visual cortex. *Proceedings: Biological Sciences*, 265(1394):359–366, 1998.

[10] A. J. Bell and T. J. Sejnowski. The "independent components" of natural scenes are edge filters. *Vision Res.*, 37(23):3327–3338, December 1997.

[11] B. Olshausen and D. Field. Sparse coding with an overcomplete basis set: A strategy employed by V1? *Nature*, 1997.

[12] A. Hyvärinen, J. Hurri, and Patrick O. Hoyer. *Natural Image Statistics: A Probabilistic Approach to Early Computational Vision. (Computational Imaging and Vision)*. Springer, 2nd printing. edition, 2009.

[13] D. J. Field. What is the goal of sensory coding? *Neural Computation*, 6(4):559–601, July 1994.

[14] B. Willmore and D. J. Tolhurst. Characterizing the sparseness of neural codes. *Network*, 12(3):255–270, January 2001.

[15] O. Schwartz and E. P. Simoncelli. Natural signal statistics and sensory gain control. *Nature Neuroscience*, 4:819–825, 2001.

[16] M.A. Ranzato, C. Poultney, S. Chopra, and Y. Lecun. Efficient learning of sparse representations with an energy-based model. In *NIPS*, 2006.

[17] A. Coates, H. Lee, and A. Y. Ng. An analysis of single-layer networks in unsupervised feature learning. In *AISTATS*, 2011.

[18] A. Treves and E. Rolls. What determines the capacity of autoassociative memories in the brain? *Network: Computation in Neural Systems*, 2:371–397(27), 1991.

[19] Y. Bengio, P. Lamblin, D. Popovici, and H. Larochelle. Greedy Layer-Wise training of deep networks. In *NIPS*, 2006.

[20] M. Schmidt. minFunc. http://www.cs.ubc.ca/~schmidtm/Software/minFunc.html, 2005.

[21] M. Ranzato and G. E. Hinton. Modeling Pixel Means and Covariances Using Factorized Third-Order Boltzmann Machines. In *CVPR*, 2010.

[22] U. Köster and A. Hyvärinen. A two-layer model of natural stimuli estimated with score matching. *Neural Computation*, 22(9):2308–2333, 2010.

[23] A. Saxe, M. Bhand, Z. Chen, P.W. Koh, B. Suresh, and A.Y. Ng. On random weights and unsupervised feature learning. In *ICML*, 2011.

[24] S. Petrov, A. Pauls, and D. Klein. Learning structured models for phone recognition. In *Proc. of EMNLP-CoNLL*, 2007.

[25] F. Sha and L.K. Saul. Large margin gaussian mixture modeling for phonetic classification and recognition. In *ICASSP*. IEEE, 2006.

[26] D. Yu, L. Deng, and A. Acero. Hidden conditional random field with distribution constraints for phone classification. In *Interspeech*, 2009.

[27] H.A. Chang and J.R. Glass. Hierarchical large-margin gaussian mixture models for phonetic classification. In *Automatic Speech Recognition & Understanding, 2007. ASRU. IEEE Workshop on*, pages 272–277. IEEE, 2007.

[28] W. E. Fisher, G. R. Doddington, and K. M. Goudle-marshall. The DARPA speech recognition research database: specifications and status. 1986.

[29] P. Clarkson and P. J. Moreno. On the use of support vector machines for phonetic classification. *Acoustics, Speech, and Signal Processing, IEEE International Conference on*, 2:585–588, 1999.

[30] J. Mairal, F. Bach, J. Ponce, and G. Sapiro. Online dictionary learning for sparse coding. In *ICML*, 2009.

[31] C.-C. Chang and C.-J. Lin. LIBSVM: A library for support vector machines. *ACM Transactions on Intelligent Systems and Technology*, 2:27:1–27:27, 2011. Software available at http://www.csie.ntu.edu.tw/~cjlin/libsvm.

[32] M. Wainwright, O. Schwartz, and E. Simoncelli. Natural image statistics and divisive normalization: Modeling nonlinearity and adaptation in cortical neurons, 2001.

[33] K. Jarrett, K. Kavukcuoglu, M. Ranzato, and Y. LeCun. What is the best multi-stage architecture for object recognition? In *ICCV*, 2009.

[34] N. Pinto, D. D. Cox, and J. J. DiCarlo. Why is Real-World visual object recognition hard? *PLoS Comput Biol*, 4(1):e27+, January 2008.

[35] Patrik O. Hoyer. Non-negative matrix factorization with sparseness constraints. *JMLR*, 5:1457–1469, 2004.

